# Learning sparse dynamic linear systems using stable spline kernels and exponential hyperpriors

**Alessandro Chiuso**
Department of Management and Engineering
University of Padova
Vicenza, Italy
alessandro.chiuso@unipd.it

**Gianluigi Pillonetto**[*]
Department of Information Engineering
University of Padova
Padova, Italy
giapi@dei.unipd.it

## Abstract

We introduce a new Bayesian nonparametric approach to identification of sparse dynamic linear systems. The impulse responses are modeled as Gaussian processes whose autocovariances encode the BIBO stability constraint, as defined by the recently introduced "Stable Spline kernel". Sparse solutions are obtained by placing exponential hyperpriors on the scale factors of such kernels. Numerical experiments regarding estimation of ARMAX models show that this technique provides a definite advantage over a group LAR algorithm and state-of-the-art parametric identification techniques based on prediction error minimization.

## 1 Introduction

Black-box identification approaches are widely used to learn dynamic models from a finite set of input/output data [1]. In particular, in this paper we focus on the identification of large scale linear systems that involve a wide amount of variables and find important applications in many different domains such as chemical engineering, economic systems and computer vision [2]. In this scenario a key point is that the identification procedure should be sparsity-favouring, i.e. able to extract from the large number of subsystems entering the system description just that subset which influences significantly the system output. Such sparsity principle permeates many well known techniques in machine learning and signal processing such as feature selection, selective shrinkage and compressed sensing [3, 4].

In the classical identification scenario, Prediction Error Methods (PEM) represent the most used approaches to optimal prediction of discrete-time systems [1]. The statistical properties of PEM (and Maximum Likelihood) methods are well understood when the model structure is assumed to be known. However, in real applications, first a set of competitive parametric models has to be postulated. Then, a key point is the selection of the most adequate model structure, usually performed by AIC and BIC criteria [5, 6]. Not surprisingly, the resulting prediction performance, when tested on experimental data, may be distant from that predicted by "standard" (i.e. without model selection) statistical theory, which suggests that PEM should be asymptotically efficient for Gaussian innovations. If this drawback may affect standard identification problems, a fortiori it renders difficult the study of large scale systems where the elevated number of parameters, as compared to the number of data available, may undermine the applicability of the theory underlying e.g. AIC and BIC.

Some novel estimation techniques inducing sparse models have been recently proposed. They include the well known Lasso [7] and Least Angle Regression (LAR) [8] where variable selection is performed exploiting the $\ell_1$ norm. This type of penalty term encodes the so called bi-separation

---

[*]This research has been partially supported by the PRIN Project "Sviluppo di nuovi metodi e algoritmi per l'identificazione, la stima Bayesiana e il controllo adattativo e distribuito", by the Progetto di Ateneo CPDA090135/09 funded by the University of Padova and by the European Community's Seventh Framework Programme under agreement n. FP7-ICT-223866-FeedNetBack.

feature, i.e. it favors solutions with many zero entries at the expense of few large components. Consistency properties of this method are discussed e.g. in [9, 10]. Extensions of this procedure for group selection include Group Lasso and Group LAR (GLAR) [11] where the sum of the Euclidean norms of each group (in place of the absolute value of the single components) is used. Theoretical analyses of these approaches and connections with the multiple kernel learning problem can be found in [12, 13]. However, most of the work has been done in the "static" scenario while very little, with some exception [14, 15], can be found regarding the identification of dynamic systems.

In this paper we adopt a Bayesian point of view to prediction and identification of sparse linear systems. Our starting point is the new identification paradigm developed in [16] that relies on nonparametric estimation of impulse responses (see also [17] for extensions to predictor estimation). Rather than postulating finite-dimensional structures for the system transfer function, e.g. ARX, ARMAX or Laguerre [1], the system impulse response is searched for within an infinite-dimensional space. The intrinsical ill-posed nature of the problem is circumvented using Bayesian regularization methods. In particular, working under the framework of Gaussian regression [18], in [16] the system impulse response is modeled as a Gaussian process whose autocovariance is the so called *stable spline kernel* that includes the BIBO stability constraint.

In this paper, we extend this nonparametric paradigm to the design of optimal linear predictors for sparse systems. Without loss of generality, analysis is restricted to MISO systems so that we interpret the predictor as a system with $m + 1$ inputs (given by past outputs and inputs) and one output (output predictions). Thus, predictor design amounts to estimating $m + 1$ impulse responses modeled as realizations of Gaussian processes. We set their autocovariances to stable spline kernels with different (and unknown) scale factors which are assigned exponential hyperpriors having a common hypervariance. In this way, while GLAR uses the sum of the $\ell_1$ norms of the single impulse responses, our approach favors sparsity through an $\ell_1$ penalty on kernel hyperparameters. Inducing sparsity by hyperpriors is an important feature of our approach. In fact, this permits to obtain the marginal posterior of the hyperparameters in closed form and hence also their estimates in a robust way. Once the kernels are selected, the impulse responses are obtained by a convex Tikhonov-type variational problem. Numerical experiments involving sparse ARMAX systems show that this approach provides a definite advantage over both GLAR and PEM (equipped with AIC or BIC) in terms of predictive capability on new output data.

The paper is organized as follows. In Section 2, the nonparametric approach to system identification introduced in [16] is briefly reviewed. Section 3 reports the statement of the predictor estimation problem while Section 4 describes the new Bayesian model for system identification of sparse linear systems. In Section 5, a numerical algorithm which returns the unknown components of the prior and the estimates of predictor and system impulse responses is derived. In Section 6 we use simulated data to demonstrate the effectiveness of the proposed approach. Conclusions end the paper.

## 2 Preliminaries: kernels for system identification

### 2.1 Kernel-based regularization

A widely used approach to reconstruct a function from indirect measurements $\{y_t\}$ consists of minimizing a regularization functional in a reproducing kernel Hilbert space (RKHS) $\mathcal{H}$ associated with a symmetric and positive-definite kernel $K$ [19]. Given $N$ data points, least-squares regularization in $\mathcal{H}$ estimates the unknown function as

$$\hat{h} = \arg\min_h \sum_{t=1}^{N} (y_t - \Gamma_t[h])^2 + \eta \|h\|_{\mathcal{H}}^2 \tag{1}$$

where $\{\Gamma_t\}$ are linear and bounded functionals on $\mathcal{H}$ related to the measurement model while the positive scalar $\eta$ trades off empirical error and solution smoothness [20].

Under the stated assumptions and according to the representer theorem [21], the minimizer of (1) is the sum of $N$ basis functions defined by the kernel filtered by the operators $\{\Gamma_t\}$, with coefficients obtainable solving a linear system of equations. Such solution enjoys also an interpretation in Bayesian terms. It corresponds to the minimum variance estimate of $f$ when $f$ is a zero-mean Gaussian process with autocovariance $K$ and $\{y_t - \Gamma_t[f]\}$ is white Gaussian noise independent of $f$ [22]. Often, prior knowledge is limited to the fact that the signal, and possibly some of its derivatives, are continuous with bounded energy. In this case, $f$ is often modeled as the $p$-fold integral of

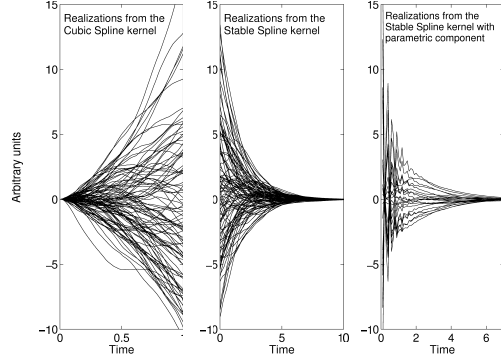

Figure 1: Realizations of a stochastic process $f$ with autocovariance proportional to the standard Cubic Spline kernel (left), the new Stable Spline kernel (middle) and its sampled version enriched by a parametric component defined by the poles $-0.5 \pm 0.6\sqrt{-1}$ (right).

white noise. If the white noise has unit intensity, the autocorrelation of $f$ is $W_p$ where

$$W_p(s,t) = \int_0^1 G_p(s,u)G_p(t,u)du, \qquad G_p(r,u) = \frac{(r-u)_+^{p-1}}{(p-1)!}, \quad (u)_+ = \begin{cases} u & \text{if } u \geq 0 \\ 0 & \text{if } u < 0 \end{cases} \quad (2)$$

This is the autocovariance associated with the Bayesian interpretation of $p$-th order smoothing splines [23]. In particular, when $p = 2$, one obtains the cubic spline kernel.

## 2.2 Kernels for system identification

In the system identification scenario, the main drawback of the kernel (2) is that it does not account for impulse response stability. In fact, the variance of $f$ increases over time. This can be easily appreciated by looking at Fig. 1 (left) which displays 100 realizations drawn from a zero-mean Gaussian process with autocovariance proportional to $W_2$. One of the key contributions of [16] is the definition of a kernel specifically suited to linear system identification leading to an estimator with favorable bias and variance properties. In particular, it is easy to see that if the autocovariance of $f$ is proportional to $W_p$, the variance of $f(t)$ is zero at $t = 0$ and tends to $\infty$ as $t$ increases. However, if $f$ represents a stable impulse response, we would rather let it have a finite variance at $t = 0$ which goes exponentially to zero as $t$ tends to $\infty$. This property can be ensured by considering autocovariances proportional to the class of kernels given by

$$K_p(s,t) = W_p(e^{-\beta s}, e^{-\beta t}), \quad s, t \in \mathbb{R}^+ \tag{3}$$

where $\beta$ is a positive scalar governing the decay rate of the variance [16]. In practice, $\beta$ will be unknown so that it is convenient to treat it as a hyperparameter to be estimated from data.

In view of (3), if $p = 2$ the autocovariance becomes the Stable Spline kernel introduced in [16]:

$$K_2(t,\tau) = \frac{e^{-\beta(t+\tau)}e^{-\beta \max(t,\tau)}}{2} - \frac{e^{-3\beta \max(t,\tau)}}{6} \tag{4}$$

**Proposition 1** *[16] Let $f$ be zero-mean Gaussian with autocovariance $K_2$. Then, with probability one, the realizations of $f$ are continuous impulse responses of BIBO stable dynamic systems.*

The effect of the stability constraint is visible in Fig. 1 (middle) which displays 100 realizations drawn from a zero-mean Gaussian process with autocovariance proportional to $K_2$ with $\beta = 0.4$.

## 3 Statement of the system identification problem

In what follows, vectors are column vectors, unless other is specified. We denote with $\{y_t\}_{t \in \mathbb{Z}}$, $y_t \in \mathbb{R}$ and $\{u_t\}_{t \in \mathbb{Z}}$, $u_t \in \mathbb{R}^m$ a pair of jointly stationary stochastic processes which represent,

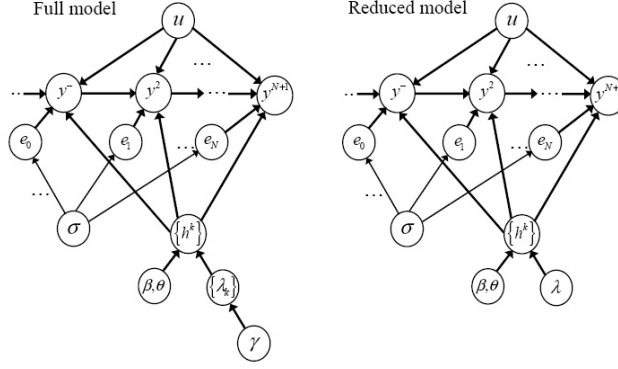

Figure 2: Bayesian network describing the new nonparametric model for identification of sparse linear systems where $y^l := [y_{l-1}, y_{l-2}, \ldots]$ and, in the reduced model, $\lambda := \lambda_1 = \ldots = \lambda_{m+1}$.

respectively, the output and input of an unknown time-invariant dynamical system. With some abuse of notation, $y_t$ will both denote a random variable (from the random process $\{y_t\}_{t \in \mathbb{Z}}$) and its sample value. The same holds for $u_t$. Our aim is to identify a linear dynamical system of the form

$$y_t = \sum_{i=1}^{\infty} f_i u_{t-i} + \sum_{i=0}^{\infty} g_i e_{t-i} \qquad (5)$$

from $\{u_t, y_t\}_{t=1,..,N}$. In (5), $f_i \in \mathbb{R}^{1 \times m}$ and $g_i \in \mathbb{R}$ are matrix and scalar coefficients of the unknown system impulse responses while $e_t$ is the Gaussian innovation sequence.

Following the Prediction Error Minimization framework, identification of the dynamical system (5) is converted in estimation of the associated one-step-ahead predictor. Letting $h^k := \{h^k_t\}_{t \in \mathbb{N}}$ denote the predictor impulse response associated with the $k$-th input $\{u^k_t\}_{t \in \mathbb{Z}}$, one has

$$y_t = \sum_{k=1}^{m} \left[ \sum_{i=1}^{\infty} h^k_i u^k_{t-i} \right] + \sum_{i=1}^{\infty} h^{m+1}_i y_{t-i} + e_t \qquad (6)$$

where $h^{m+1} := \{h^{m+1}_t\}_{t \in \mathbb{N}}$ is the impulse response modeling the autoregressive component of the predictor. As is well known, if the joint spectrum of $\{y_t\}$ and $\{u_t\}$ is bounded away from zero, each $h^k$ is (BIBO) stable. Under such assumption, our aim is to estimate the predictor impulse responses, in a scenario where the number of measurements $N$ is not large, as compared with $m$, and many measured inputs could be irrelevant for the prediction of $y_t$. We will focus on the identification of ARMAX models, so that the zeta-transforms of $\{h^k\}$ are rational functions all sharing the same denominator, even if the approach described below immediately extends to general linear systems.

# 4 A Bayesian model for identification of sparse linear systems

## 4.1 Prior for predictor impulse responses

We model $\{h^k\}$ as independent Gaussian processes whose kernels share the same hyperparameters apart from the scale factors. In particular, each $h^k$ is proportional to the convolution of a zero-mean Gaussian process, with autocovariance given by the sampled version of $K_2$, with a parametric impulse response $r$, used to capture dynamics hardly represented by a smooth process, e.g. high-frequency oscillations. For instance, the zeta-transform $R(z)$ of $r$ can be parametrized as follows

$$R(z) = \frac{z^2}{P_\theta(z)}, \qquad P_\theta(z) = z^2 + \theta_1 z + \theta_2, \qquad \theta \in \Theta \subset \mathbb{R}^2 \qquad (7)$$

where the feasible region $\Theta$ constraints the two roots of $P_\theta(z)$ to belong to the open left unit semicircle in the complex plane. To better appreciate the role of the finite-dimensional component of the model, Fig. 1 (right panel) shows some realizations (with samples linearly interpolated) drawn from a discrete-time zero-mean normal process with autocovariance given by $K_2$ enriched by $\theta = [1 \quad 0.61]$ in (7). Notice that, in this way, an oscillatory behavior is introduced in the realizations

by enriching the Stable Spline kernel with the poles $-0.5 \pm 0.6\sqrt{-1}$.

The kernel of $h^k$ defined by $K_2$ and (7) is denoted by $K : \mathbb{N} \times \mathbb{N} \mapsto \mathbb{R}$ and depends on $\beta, \theta$. Thus, letting $\mathbb{E}[\cdot]$ denote the expectation operator, the prior model on the impulse responses is given by

$$\mathbb{E}[h_j^k h_i^k] = \lambda_k^2 K(j, i; \theta, \beta), \quad k = 1, \ldots, m+1, \quad i, j \in \mathbb{N}$$

### 4.2 Hyperprior for the hyperparameters

The noise variance $\sigma^2$ will always be estimated via a preliminary step using a low-bias ARX model, as described in [24]. Thus, this parameter will be assumed known in the description of our Bayesian model. The hyperparameters $\beta$, $\theta$ and $\{\lambda_k\}$ are instead modeled as mutually independent random vectors. $\beta$ is given a non informative probability density on $\mathbb{R}^+$ while $\theta$ has a uniform distribution on $\Theta$. Each $\lambda_k$ is an exponential random variable with inverse of the mean (and SD) $\gamma \in \mathbb{R}^+$, i.e.

$$\mathbf{p}(\lambda_k) = \gamma \exp\left(-\gamma \lambda_k\right) \chi(\lambda_k \geq 0), \qquad k = 1, \ldots, m+1$$

with $\chi$ the indicator function. We also interpret $\gamma$ as a random variable with a non informative prior on $\mathbb{R}^+$. Finally, $\zeta$ indicates the hyperparameter random vector, i.e. $\zeta := [\lambda_1, \ldots, \lambda_{m+1}, \theta_1, \theta_2, \beta, \gamma]$.

### 4.3 The full Bayesian model

Let $A_k \in \mathbb{R}^{N \times \infty}$ where, for $j = 1, \ldots, N$ and $i \in \mathbb{N}$, we have:

$$[A_k]_{ji} = u_{j-i}^k \quad \text{for} \quad k = 1, \ldots, m, \qquad [A_{m+1}]_{ji} = y_{j-i} \tag{8}$$

In view of (6), using notation of ordinary algebra to handle infinite-dimensional objects with each $h^k$ interpreted as an infinite-dimensional column vector, it holds that

$$y^+ = \sum_{k=1}^{m} A_k(u^k) h^k + A_{m+1}(y^+, y^-) h^{m+1} + e \tag{9}$$

$$\text{where} \quad y^+ = [y_1, y_2, \ldots, y_N]^T, \quad y^- = [y_0, y_{-1}, y_{-2}, \ldots]^T, \quad e = [e_1, e_2, \ldots, e_N]^T \tag{10}$$

In practice, $y^-$ is never completely known and a solution is to set its unknown components to zero, see e.g. Section 3.2 in [1]. Further, the following approximation is exploited:

$$\mathbf{p}(y^+, \{h^k\}, y^- | \zeta) \approx \mathbf{p}(y^+ | \{h^k\}, y^-, \zeta) \mathbf{p}(\{h^k\} | \zeta) \mathbf{p}(y^-) \tag{11}$$

i.e. the past $y^-$ is assumed not to carry information on the predictor impulse responses and the hyperparameters. Our stochastic model is described by the Bayesian network in Fig. 2 (left side).

The dependence on $y^-$ is hereafter omitted as well as dependence of the $\{A_k\}$ on $y^+$ or $u^k$. We start reporting a preliminary lemma, whose proof can be found in [17], which will be needed in propositions 2 and 3.

**Lemma 1** *Let the roots of $P_\theta$ in (7) be stable. Then, if $\{y_t\}$ and $\{u_t\}$ are zero mean, finite variance stationary stochastic processes, each operator $\{A_k\}$ is almost surely (a.s.) continuous in $\mathcal{H}_K$.*

## 5 Estimation of the hyper-parameters and the predictor impulse responses

### 5.1 Estimation of the hyper-parameters

We estimate the hyperparameter vector $\zeta$ by optimizing its marginal posterior, i.e. the joint density of $y^+, \zeta$ and $\{h^k\}$ where all the $\{h^k\}$ are integrated out. This is described in the next proposition that derives from simple manipulations of probability densities whose well-posedness is guaranteed by lemma 1. Below, $I_N$ is the $N \times N$ identity matrix while, with a slight abuse of notation, $K$ is now seen as an element of $\mathbb{R}^{\infty \times \infty}$, i.e. its $i$-th column is the sequence $K(\cdot, i), i \in \mathbb{N}$.

**Proposition 2** *Let $\{y_t\}$ and $\{u_t\}$ be zero mean, finite variance stationary stochastic processes. Then, under the approximation (11), the maximum a posteriori estimate of $\zeta$ given $y^+$ is*

$$\hat{\zeta} = \arg\min_{\zeta} J(y^+; \zeta) \quad s.t. \quad \theta \in \Theta, \quad \gamma, \beta > 0, \quad \lambda_k \geq 0 \quad (k = 1, \ldots, m+1) \tag{12}$$

*where $J$ is almost surely well defined pointwise and given by*

$$J(y^+; \zeta) = \frac{1}{2} \log \left( \det[2\pi V[y^+]] \right) + \frac{1}{2}(y^+)^T (V[y^+])^{-1} y^+ + \gamma \sum_{k=1}^{m+1} \lambda_k - \log(\gamma) \qquad (13)$$

*with $V[y^+] = \sigma^2 I_N + \sum_{k=1}^{m+1} \lambda_k A_k K A_k^T$.*

The objective (13), including the $\ell_1$ penalty on $\{\lambda_k\}$, is a Bayesian modified version of that connected with multiple kernel learning, see Section 3 in [25]. Additional terms are $\log \left( \det[V[y^+]] \right)$ and $\log(\gamma)$ that permits to estimate the weight of the $\ell_1$ norm jointly with the other hyperparameters. An important issue for the practical use of our numerical scheme is the availability of a good starting point for the optimizer. Below, we describe a scheme that achieves a suboptimal solution just solving an optimization problem in $\mathbb{R}^4$ related to the reduced Bayesian model of Fig. 2 (right side).

i) Obtain $\{\hat{\lambda}_k\}$, $\hat{\theta}$ and $\hat{\beta}$ solving the following modified version of problem (12)

$$\arg \min_{\zeta} \left[ J(y^+; \zeta) - \gamma \sum_{k=1}^{m+1} \lambda_k + \log(\gamma) \right] \text{ s.t. } \theta \in \Theta, \quad \beta > 0, \quad \lambda_1 = \ldots = \lambda_{m+1} \geq 0$$

ii) Set $\hat{\gamma} = 1/\hat{\lambda}_1$ and $\hat{\zeta} = [\hat{\lambda}_1, \ldots, \hat{\lambda}_{m+1}, \hat{\theta}, \hat{\beta}, \hat{\gamma}]$. Then, for $k = 1, \ldots, m+1$: set $\bar{\zeta} = \hat{\zeta}$ except for the $k$-th component of $\bar{\zeta}$ which is set to 0; if $J(y^+; \bar{\zeta}) \leq J(y^+; \hat{\zeta})$, set $\hat{\zeta} = \bar{\zeta}$.

## 5.2  Estimation of the predictor impulse responses for known $\zeta$

Let $\mathcal{H}_K$ be the RKHS associated with $K$, with norm $\| \cdot \|_{\mathcal{H}_K}$. Let also $\hat{h}^k = \mathbb{E}[h^k | y^+, \zeta]$. The following result comes from the representer theorem whose applicability is guaranteed by lemma 1.

**Proposition 3** *Under the same assumptions of Proposition 2, almost surely we have*

$$\{\hat{h}^k\}_{k=1}^{m+1} = \arg \min_{\{f^k \in \mathcal{H}_K\}_{k=1}^{m+1}} \|y^+ - \sum_{k=1}^{m+1} A_k f^k\|^2 + \sigma^2 \sum_{k=1}^{m+1} \frac{\|f^k\|_{\mathcal{H}_K}^2}{\lambda_k^2}$$

*where $\| \cdot \|$ is the Euclidean norm. Moreover, almost surely we also have for $k = 1, \ldots, m+1$*

$$\hat{h}^k = \lambda_k^2 K A_k^T c, \qquad c = \left( \sigma^2 I_N + \sum_{k=1}^{m+1} \lambda_k A_k K A_k^T \right)^{-1} y^+ \qquad (14)$$

After obtaining the estimates of the $\{h^k\}$, simple formulas can then be used to derive the system impulse responses $f$ and $g$ in (5) and hence also the $k$-step ahead predictors, see [1] for details.

## 6  Numerical experiments

We consider two Monte Carlo studies of 200 runs where at any run an ARMAX linear system with 15 inputs is generated as follows

- the number of $h^k$ different from zero is randomly drawn from the set $\{0, 1, 2, .., 8\}$.
- Then, the order of the ARMAX model is randomly chosen in $[1, 30]$ and the model is generated by the MATLAB function `drmodel.m`. The system and the predictor poles are restricted to have modulus less than $0.95$ with the $\ell_2$ norm of each $h^k$ bounded by 10.

In the first Monte Carlo experiment, at any run an identification data set of size 500 and a test set of size 1000 is generated using independent realizations of white noise as input. In the second experiment, the prediction on new data is more challenging. In fact, at any run, an identification data set of size 500 and a test set of size 1000 is generated via the MATLAB function `idinput.m` using, respectively, independent realizations of a random Gaussian signal with band $[0, 0.8]$ and $[0, 0.9]$ (the interval boundaries specify the lower and upper limits of the passband, expressed as fractions of the Nyquist frequency). We compare the following estimators:

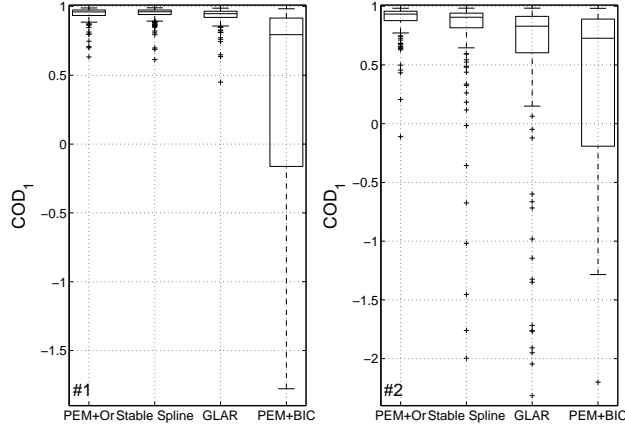

Figure 3: Boxplots of the values of $COD_1$ obtained by PEM+Or, Stable Spline, GLAR and PEM+BIC in the two experiments. The outliers obtained by PEM+BIC are not all displayed.

| Experiment | PEM+Oracle | Stable Spline | Subopt. Stable Spline | GLAR |
|:---:|:---:|:---:|:---:|:---:|
| #1 | 100% | 98.7% | 97.5% | 45.6% |
| #2 | 100% | 98.4% | 98.2% | 52.4% |

Table 1: Percentage of the $h^k$ equal to zero correctly set to zero by the employed estimator.

1. GLAR: this is the GLAR algorithm described in [11] applied to ARX models; the order (between 1 and 30) and the level of sparsity (i.e. the number of null $h^k$) is determined using the first $2/3$ of the 500 available data as training set and the remaining part as validation data (the use of $C_p$ statistics does not provide better results in this case).

2. PEM+Oracle: this is the classical PEM approach, as implemented in the pem.m function of the MATLAB System Identification Toolbox [26], equipped with an oracle that, at every run, knows which predictor impulse response are zero and, having access to the test set, selects those model orders that provide the best prediction performance.

3. PEM+BIC: this is the classical PEM approach that uses BIC for model order selection. The order of the polynomials in the ARMAX model are not allowed to be different each other since this would lead to a combinatorial explosion of the number of competitive models.

4. Stable Spline: this is the approach based on the full Bayesian model of Fig. 2. The first 40 available input/output pairs enter the $\{A_k\}$ in (9) so that $N = 460$. For computational reasons, the number of estimated predictor coefficients is 40.

5. Suboptimal Stable Spline: the same as above except that we exploit the reduced Bayesian model of Fig. 2 complemented with the procedure described at the end of subsection 5.1.

The following performance indexes are considered:

1. Percentage of the impulse responses equal to zero correctly set to zero by the estimator.

2. $k$-step-ahead Coefficient of Determination, denoted by $COD_k$, quantifying how much of the test set variance is explained by the forecast. It is computed at each run as

$$COD_k := 1 - \frac{RMS_k^2}{\frac{1}{1000}\sum_{i=1}^{1000}(y_t^{test} - \bar{y}_t^{test})^2}, \qquad RMS_k := \sqrt{\frac{1}{1000}\sum_{t=1}^{1000}(y_t^{test} - \hat{y}_{t|t-k}^{test})^2}$$

$$(15)$$

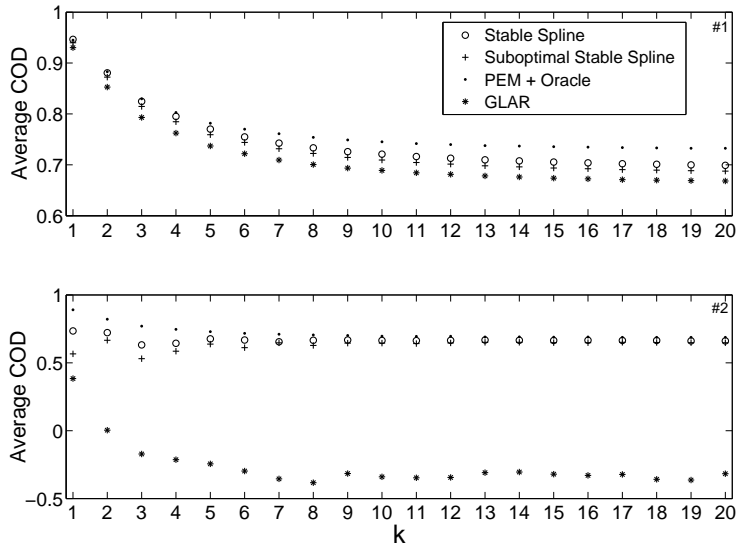

Figure 4: $\overline{COD}_k$, i.e. average coefficient of determination relative to $k$-step ahead prediction, obtained during the Monte Carlo study #1 (top) and #2 (bottom) using PEM+Oracle (●), GLAR (∗) Stable Spline based on the full (○) and the reduced (+) Bayesian model of Fig. 2.

where $\bar{y}^{test}$ is the sample mean of the test set data $\{y_t^{test}\}_{t=1}^{1000}$ and $\hat{y}_{t|t-k}^{test}$ is the $k$-step ahead prediction computed using the estimated model. The average index obtained during the Monte Carlo study, as a function of $k$, is then denoted by $\overline{COD}_k$.

Notice that, in both of the cases, the larger the index, the better is the performance of the estimator. In every experiment the performance of PEM+BIC has been largely unsatisfactory, providing strongly negative values for $\overline{COD}_k$. This is illustrated e.g. in Fig. 3 showing the boxplots of the 200 values of $COD_1$ obtained by 4 of the employed estimators during the two Monte Carlo studies. We have also assessed that results do not improve using AIC. In view of this, in what follows other results from PEM+BIC will not be shown.

Table 1 reports the percentage of the predictor impulse responses equal to zero correctly estimated as zero by the estimators. Remarkably, in all the cases the Stable Spline estimators not only outperform GLAR but the achieved percentage is close to 99%. This shows that the use of the marginal posterior permits to effectively detect the subset of the $\{\lambda_k\}$ equal to zero. Finally, Fig. 4 displays $\overline{COD}_k$ as a function of the prediction horizon obtained during the Monte Carlo study #1 (top) and #2 (bottom). The performance of Stable Spline appears superior than that of GLAR and is comparable with that of PEM+Oracle also when the reduced Bayesian model of Fig. 2 is used.

## 7 Conclusions

We have shown how identification of large sparse dynamic systems can benefit from the flexibility of kernel methods. To this aim, we have extended a recently proposed nonparametric paradigm to identify sparse models via prediction error minimization. Predictor impulse responses are modeled as zero-mean Gaussian processes using stable spline kernels encoding the BIBO-stability constraint and sparsity is induced by exponential hyperpriors on their scale factors. The method compares much favorably with GLAR, with its performance close to that achievable combining PEM with an oracle which exploits the test set in order to select the best model order. In the near future we plan to provide a theoretical analysis characterizing the hyperprior-based scheme as well as to design new ad hoc optimization schemes for hyperparameters estimation.

# References

[1] L. Ljung. *System Identification - Theory For the User*. Prentice Hall, 1999.

[2] J. Mohammadpour and K.M. Grigoriadis. *Efficient Modeling and Control of Large-scale Systems*. Springer, 2010.

[3] T. J. Hastie and R. J. Tibshirani. Generalized additive models. In *Monographs on Statistics and Applied Probability*, volume 43. Chapman and Hall, London, UK, 1990.

[4] D. Donoho. Compressed sensing. *IEEE Trans. on Information Theory*, 52(4):1289–1306, 2006.

[5] H. Akaike. A new look at the statistical model identification. *IEEE Transactions on Automatic Control*, 19:716–723, 1974.

[6] G. Schwarz. Estimating the dimension of a model. *The Annals of Statistics*, 6:461–464, 1978.

[7] R. Tibshirani. Regression shrinkage and selection via the LASSO. *Journal of the Royal Statistical Society, Series B.*, 58, 1996.

[8] B. Efron, T. Hastie, L. Johnstone, and R. Tibshirani. Least angle regression. *Annals of Statistics*, 32:407–499, 2004.

[9] P. Zhao and B. Yu. On model selection consistency of lasso. *Journal of Machine Learning Research*, 7:2541–2563, 2006.

[10] H. Zou. The adaptive lasso and its oracle properties. *Journal of the American Statistical Association*, 101:1418–1429, 2006.

[11] Ming Yuan and Yi Lin. Model selection and estimation in regression with grouped variables. *Journal of the Royal Statistical Society, Series B*, 68:49–67, 2006.

[12] F.R. Bach. Consistency of the group lasso and multiple kernel learning. *J. Mach. Learn. Res.*, 9:1179–1225, 2008.

[13] C. A. Micchelli and M. Pontil. Learning the kernel function via regularization. *Journal of Machine Learning Research*, 6:1099–1125, 2005.

[14] H. Wang, G. Li, and C.L. Tsai. Regression coefficient and autoregressive order shrinkage and selection via the lasso. *Journal Of The Royal Statistical Society Series B*, 69(1):63–78, 2007.

[15] Nan-Jung Hsu, Hung-Lin Hung, and Ya-Mei Chang. Subset selection for vector autoregressive processes using lasso. *Computational Statistics and Data Analysis*, 52:3645–3657, 2008.

[16] G. Pillonetto and G. De Nicolao. A new kernel-based approach for linear system identification. *Automatica*, 46(1):81–93, 2010.

[17] G. Pillonetto, A. Chiuso, and G. De Nicolao. Prediction error identification of linear systems: a nonparametric Gaussian regression approach. *Automatica (in press)*, 2011.

[18] C.E. Rasmussen and C.K.I. Williams. *Gaussian Processes for Machine Learning*. The MIT Press, 2006.

[19] N. Aronszajn. Theory of reproducing kernels. *Transactions of the American Mathematical Society*, 68:337–404, 1950.

[20] G. Wahba. Support vector machines, reproducing kernel Hilbert spaces and randomized GACV. Technical Report 984, Department of Statistics, University of Wisconsin, 1998.

[21] G. Kimeldorf and G. Wahba. Some results on Tchebycheffian spline functions. *Journal of Mathematical Analysis and Applications*, 33(1):82–95, 1971.

[22] A. J. Smola and B. Schölkopf. Bayesian kernel methods. In S. Mendelson and A. J. Smola, editors, *Machine Learning, Proceedings of the Summer School, Australian National University*, pages 65–117, Berlin, Germany, 2003. Springer-Verlag.

[23] G. Wahba. *Spline models for observational data*. SIAM, Philadelphia, 1990.

[24] G.C. Goodwin, M. Gevers, and B. Ninness. Quantifying the error in estimated transfer functions with application to model order selection. *IEEE Transactions on Automatic Control*, 37(7):913–928, 1992.

[25] F. Dinuzzo. Kernel machines with two layers and multiple kernel learning. Technical report, Preprint arXiv:1001.2709, 2010. Available at http://www-dimat.unipv.it/ dinuzzo.

[26] L. Ljung. *System Identification Toolbox V7.1 for Matlab*. Natick, MA: The MathWorks, Inc., 2007.

